# Multiplicative Forests for Continuous-Time Processes

**Jeremy C. Weiss**
University of Wisconsin
Madison, WI 53706, USA
jcweiss@cs.wisc.edu

**Sriraam Natarajan**
Wake Forest University
Winston Salem, NC 27157, USA
snataraj@wakehealth.edu

**David Page**
University of Wisconsin
Madison, WI 53706, USA
page@biostat.wisc.edu

## Abstract

Learning temporal dependencies between variables over continuous time is an important and challenging task. Continuous-time Bayesian networks effectively model such processes but are limited by the number of conditional intensity matrices, which grows exponentially in the number of parents per variable. We develop a partition-based representation using regression trees and forests whose parameter spaces grow linearly in the number of node splits. Using a multiplicative assumption we show how to update the forest likelihood in closed form, producing efficient model updates. Our results show multiplicative forests can be learned from few temporal trajectories with large gains in performance and scalability.

## 1 Introduction

The modeling of temporal dependencies is an important and challenging task with applications in fields that use forecasting or retrospective analysis, such as finance, biomedicine, and anomaly detection. While analyses over time series data with fixed, discrete time intervals are well studied, as for example in [1], there are domains in which discretizing the time leads to intervals where no observations are made, producing "missing data" in those periods, or there is no natural discretization available and so the time series assumptions are restrictive. Of note, experiments in previous work provide evidence that coercing continuous-time data into time series and conducting time series analysis is less effective than learning models built with continuous-time data in mind [2].

We investigate a subset of continuous-time models: probabilistic models over finite event spaces across continuous time. The prevailing model in this field is the continuous-time Markov process (CTMP), a model that provides an initial distribution over states and a rate matrix parameterizing the rate of transitioning between states. However, this model does not scale for the case where a CTMP state is a joint state over many variable states. Because the number of joint states is exponential in the number of variables, the size of the CTMP rate matrix grows exponentially in the number of variables. Continuous-time Bayesian networks (CTBNs), a family of CTMPs with a factored representation, encode rate matrices for each variable and the dependencies among variables [3]. Figure 1 shows a complete trajectory, i.e., a timeline where the state of each variable is known for all times $t$, for a CTMP with four joint states $(a, b)$, $(a, B)$, $(A, b)$, and $(A, B)$ factorized into two binary CTBN variables $\alpha$ and $\beta$ (with states $a$ and $A$, and $b$ and $B$, respectively).

Previous work on CTBNs includes several approaches to performing CTBN inference [4, 5, 6, 7, 8] and learning [2, 3]. Briefly, CTBNs do not admit exact inference without transformation to the exponential-size CTMP. Approximate inference methods including expectation propagation [4], mean field [6], importance sampling-based methods [7], and MCMC [8] have been applied, and while these methods have helped mitigate the inference problem, inference in large networks remains a challenge. CTBN learning involves parameter learning using sufficient statistics (e.g. numbers of transitions $M$ and durations $T$ in Figure 1) and structure learning over a directed (possibly cyclic) graph over the variables to maximize a penalized likelihood score. Our work addresses learning in a generalized framework to which the inference methods mentioned above can be extended.

In this work we introduce a generalization of CTBNs: partition-based CTBNs. Partition-based CTBNs remove the restriction used in CTBNs of storing one rate matrix per parents setting for every variable. Instead partition-based CTBNs define partitions over the joint state space and define the transition rate of each variable to be dependent on the membership of the current joint state to an element (part) of a partition. As an example, suppose we have partition $P$ composed of parts $p_1 = \{(a, b), (A, b)\}$ and $p_2 = \{(a, B), (A, B)\}$. Then the transition into $s_i$ from joint state $(A, B)$ in Figure 1 would be parameterized by transition rate $q_{a|p_2}$. Partition-based CTBNs store one transition rate per part, as opposed to one transition rate matrix per parents setting. Later we will show that, for a particular choice of partitions, a partition-based CTBN is equivalent to a CTBN. However, the more general framework offers other choices of partitions which may be more suitable for learning from data.

Partition-based CTBNs avoid one limitation of CTBNs: that the model size is necessarily exponential in the maximum number of parents per variable. For networks with sparse incoming connections, this issue is not apparent. However, in many real domains, a variable's transition rate may be a function of many variables.

Given the framework of partition-based CTBNs, we need to provide a way to determine useful partitions. Thus, we introduce partition-based CTBN learning using regression tree modifications in place of CTBN learning using graph operators of adding, reversing, and deleting edges. In the spirit of context-specific independence [9], we can view tree learning as a method for learning compact partition-based dependencies. However, tree learning induces recursive subpartitions, which limits their ability to partition the joint state space. We therefore introduce multiplicative forests for CTBNs, which allow the model to represent up to an exponential number of transition rates with parameters still linear in the number of splits.

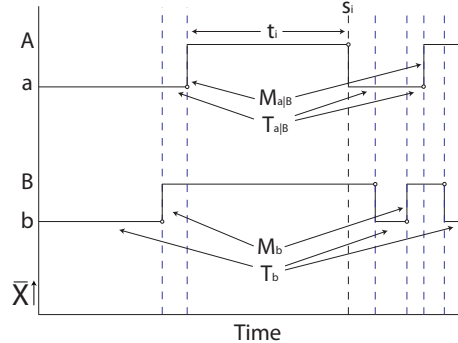

Figure 1: Example of a complete trajectory in a two-node CTBN. The arrows show the transitions and time intervals that are aggregated to compute selected sufficient statistics (M's and T's). $A$ and $a$ denote two states for one variable, and $B$ and $b$ two states for a second variable.

Following canonical tree learning methods, we perform greedy tree and forest learning using iterative structure modifications. We show that the partition-based change in log likelihood can be calculated efficiently in closed form using a multiplicative assumption. We also show that using multiplicative forests, we can efficiently calculate the ML parameters. Thus, we can calculate the maximum change in log likelihood for a forest modification proposal, which gives us the best iterative update to the forest model.

Finally, we conduct experiments to compare CTBNs, regression tree CTBNs (treeCTBNs) and multiplicative forest CTBNs (mfCTBNs) on three data sets. Our hypothesis is twofold: first, that learning treeCTBNs and mfCTBNs will scale better towards large domains because of their compact model structures, and second, that mfCTBNs will outperform both CTBNs and treeCTBNs with fewer data points because of their ability to capture multiplicative dependencies.

The rest of the paper is organized as follows: in Section 2 we provide background on CTBNs. In Section 3 we present partition-based CTBNs, show that they subsume CTBNs and define the partitions that tree and forest structures induce. We also describe theoretical advantages of using forests for learning and how to learn these models efficiently. We present results in Section 4 showing that forest CTBNs are scalable to large state spaces and learn better than CTBNs, from fewer examples and in less time. Finally, in Sections 5 and 6 we identify connections to functional gradient boosting and related continuous-time processes and discuss how our work addresses one limitation that prevents CTBNs from finding widespread use.

## 2  Background

CTBNs are probabilistic graphical models that capture dependencies between variables over continuous time. A CTBN is defined by 1) a distribution for the initial state over variables $\mathcal{X}$ given by a Bayesian Network $\mathcal{B}$, and 2) a directed (possibly cyclic) graph over variables $\mathcal{X}$ with a set of *Conditional Intensity Matrices* (CIMs) for each variable $X \in \mathcal{X}$ that hold the rates (intensities) $q_{x|u}$ of variable transitions given their parents $U_X$ in the directed graph. Here a CTBN variable $X \in \mathcal{X}$ has states $x^1, \dots, x^k$, and there is an intensity $q_{x|u}$ for every state $x \in X$ given an instantiation over its parents $u \in U_X$. The intensity corresponds to the rate of transitioning out of state $x$; the probability density function for staying in state $x$ given an instantiation of parents $u$ is $q_{x|u}e^{-q_{x|u}t}$. Given a transition, $X$ moves to some other state $x'$ with probability $\Theta_{xx'|u}$. Taking the product over intervals bounded by single transitions, we obtain the CTBN trajectory likelihood:

$$\prod_{X \in \mathcal{X}} \prod_{x \in X} \prod_{u \in U_X} q_{x|u}^{M_{x|u}} e^{-q_{x|u}T_{x|u}} \prod_{x' \neq x} \Theta_{xx'|u}^{M_{xx'|u}}$$

where the $M_{x|u}$ and $M_{xx'|u}$ are the sufficient statistics indicating the number of transitions out of state $x$ (total, and to $x'$, respectively), and the $T_{x|u}$ are the sufficient statistics for the amount of time spent in $x$ given the parents are in state $u$.

## 3  Partition-based CTBNs

Here we define partition-based CTBNs, an alternative framework for determining variable transition rates. We give the syntax and semantics of our model, providing the generative model and likelihood formulation. We then show that CTBNs are one instance in our framework. Next, we introduce regression trees and multiplicative forests and describe the partitions they induce, which are then used in the partition-based CTBN framework. Finally, we discuss the advantages of using trees and forests in terms of learning compact models efficiently.

Let $\mathcal{X}$ be a finite set of discrete variables $X$ of size $n$, with each variable $X$ having a discrete set of states $\{x^1, x^2, \dots, x^k\}$, where $k$ may differ for each variable. We define a joint state $s = \{x_1, x_2, \dots, x_n\}$ over $\mathcal{X}$ where the subscript indicates the variable index. We also define the partition space $\mathcal{P} = \mathcal{X}^1$. We will shortly define set partitions $P$ over $\mathcal{P}$, composed of disjoint parts $p$, each of which holds a set of elements $s$.

Next we define the dynamics of the model, which form a continuous-time process over $\mathcal{X}$. Each variable $X$ transitions among its states with rate parameter $q_{x'|s}$ for entering state $x'$ given the joint state $s^2$. This rate parameter (called an intensity) parameterizes the exponential distribution for transitioning into $x'$, given by the pdf: $p(x', s, t) = q_{x'|s}e^{-q_{x'|s}t}$ for time $t \in [0, \infty)$.

A partition-based CTBN has a collection of set partitions $P$ over $\mathcal{P}$, one $P_{x'}$ for every variable state $x'$. For shorthand, we will often denote $p = P_{x'}(s)$ to indicate the part $p$ of partition $P_{x'}$ to which state $s$ belongs. We define the intensity parameter as $q_{x'|s} = q_{x'|p}$ for all $s \in p$. Note that this fixes this intensity to be the same for every $s \in p$, and also note that the set of parts $p$ covers $\mathcal{P}$. The pdf for transitioning is given by $p(x', s, t) = p(x', P_{x'}(s), t) = q_{x'|p}e^{-q_{x'|p}t}$ for all $s$ in $p$.

Now we are ready to define the partition-based CTBN model. A partition-based CTBN model $\mathcal{M}$ is composed of a distribution over the initial state of our variables, defined by a Bayesian network $\mathcal{B}$, and a set of partitions $P_{x'}$ for every variable state $x'$ with corresponding sets of intensities $q_{x'|p}$.

The partition-based CTBN provides a generative framework for producing a trajectory $z$ defined by a sequence of (state, time) pairs $(s_i, t_i)$. Given an initial state $s_0$, transition times are sampled for each variable state $x'$ according to $p(x', P_{x'}(s_0), t)$. The next state is selected based on the transition to the $x'$ with the shortest time, after which the transition times are resampled according to $p(x', s_i, t)$. Due to the memoryless property of exponential distributions, no resampling of the transition time for $x'$ is needed if $p(x', s_i, t) = p(x', s_{i-1}, t)$. The trajectory terminates when all sampled transition times exceed a specified ending time.

Given a trajectory $z$, we can also define the model likelihood. For each interval $t_i$, the joint state remains unchanged, and then one variable transitions into $x'$. The likelihood given the interval is: $q_{x'|s_{i-1}} \prod_X \prod_{x \in X} e^{-q_{x|s_{i-1}} t_i}$, i.e., the product of the probability density for $x'$ and the probability that no other variable transitions before $t_i$. Taking the product over all intervals in $z$, we get the model likelihood:

$$\prod_{X \in \mathcal{X}} \prod_{x' \in X} \prod_s q_{x'|s}^{M_{x'|s}} e^{-q_{x'|s} T_s} \tag{1}$$

where $M_{x'|s}$ is the number of transitions into $x'$ from state $s$, and $T_s$ is the total duration spent in $s$. Combining terms based on the membership of $s$ to $p$ and defining $M_{x'|p} = \sum_{s \in p} M_{x'|s}$ and $T_p = \sum_{s \in p} T_s$, we get:

$$\text{Eq.(1)} = \prod_{X \in \mathcal{X}} \prod_{x' \in X} \prod_{p \in P_{x'}} q_{x'|p}^{M_{x'|p}} e^{-q_{x'|p} T_p}$$

## 3.1 CTBN as a partition-based CTBN

Here we show that CTBNs can be viewed as an instance of partition-based CTBNs. Each variable $X$ is given a parent set $U_X$, and the transition intensities $q_{x|u}$ are recorded for *leaving* donor states $x$ given the current setting of the parents $u \in U_X$. The CTBN likelihood can be shown to be:

$$\prod_{X \in \mathcal{X}} \prod_{x \in X} \prod_{u \in U_X} e^{-q_{x|u} T_{x|u}} \prod_{x' \neq x} q_{xx'|u}^{M_{xx'|u}} \tag{2}$$

as in [5], where $q_{xx'|u}$ and $M_{xx'|u}$ denote the intensity and number of transitions from state $x$ to state $x'$ given parents setting $u$, and $\sum_{x' \neq x} q_{xx'|u} = q_{x|u}$. Rearranging the product from equation 2, we achieve a likelihood in terms of recipient states $x'$:

$$\text{Eq. (2)} = \prod_{X \in \mathcal{X}} \prod_{x \in X} \prod_{u \in U_X} \prod_{x' \neq x} q_{xx'|u}^{M_{xx'|u}} e^{-q_{xx'|u} T_{x|u}}$$

$$= \prod_{X \in \mathcal{X}} \prod_{x' \in X} \prod_{p \in P_{x'}} q_{x'|p}^{M_{x'|p}} e^{-q_{x'|p} T_p} \tag{3}$$

where we define $p$ as $\{x\} \times \{u\} \times (\mathcal{X} \setminus (X \times U_X))$ in each partition $P_{x'}$, and likewise: $q_{x'|p} = q_{xx'|u}$, $M_{x'|p} = M_{xx'|u}$, and $T_p = T_{x|u}$. Thus, CTBNs are one instance of partition-based CTBNs, with partitions corresponding to a specified donor state $x$ and parents setting $u$.

## 3.2 Tree and forest partitions

Trees and forests induce partitions over a space defined by the set of possible split criteria [11]. Here we will define the Conditional Intensity Trees (CITs): regression trees that determine the intensities $q_{x'|p}$ by inducing a partition over $\mathcal{P}$. Similarly, we will define Conditional Intensity Forests (CIFs), where tree intensities are named intensity factors whose product determines $q_{x'|p}$. An example of a CIF, composed of a collection of CITs, is shown later in the experiment results in Figure 4.

Formally, a *Conditional Intensity Tree* (CIT) $f_{x'}$ is a directed tree structure on a graph $G(V, E)$ with nodes $V$ and edges $E(V_i, V_j)$. Internal nodes $V_i$ of the tree hold splits $\sigma_{V_i} = (\pi_{V_i}, \{E(V_i, \cdot)\})$ composed of surjective maps $\pi_{V_i} : s \mapsto E(V_i, V_j)$ and lists of the outgoing edges. The maps $\pi$ induce partitions over $\mathcal{P}$ and endow each outgoing edge $E(V_i, V_j)$ with part $p_{V_j}$. External nodes $l$, or leaves, hold non-negative real values $q_{x'|p}^{\text{CIT}}$ called intensities. A path $\rho$ from the root to a leaf induces a part $p$, which is the intersection of the parts on the edges of the path: $p = \bigcap_{E(V_i, V_j) \in \rho} p_{V_j}$. The parts corresponding to paths of a CIT form a partition over $\mathcal{P}$, which can be shown easily using induction and the fact that the maps $\pi_{V_i}$ induce disjoint parts $p_{V_j}$ that cover $\mathcal{P}$.

A *Conditional Intensity Forest* (CIF) $\mathcal{F}_{x'}$ is a set of CITs $\{f_{x'}\}$. Because the parts of each CIT form a partition, a CIF induces a joint partition over $\mathcal{P}$ where a part $p$ is the set of states $s$ that have the same paths through all CITs. Finally, a CIF produces intensities from joint states by taking the product over the intensity factors from each CIT: $q_{x'|p^{\text{CIF}}}^{\text{CIF}} = \prod_{f_{x'}} q_{x'|p^{\text{CIT}}}^{\text{CIT}}$.

Using regression trees and forests can greatly reduce the number of model parameters. In CTBNs, the number of parameters grows exponentially in the number of parents per node. In tree and forest CTBNs, the number of parameters may be linear in the number of parents per node, exploiting the efficiency of using partitions. Notably, however, tree CTBNs are limited to having one intensity per parameter. In forest CTBNs, the number of intensities can be exponential in the number of parameters. Thus, the forest model has much greater potential expressivity per parameter than the other models. We quantify these differences in the Supplementary Materials at our website.

### 3.3 Forest CTBN learning

Here we discuss the reasoning for using the multiplicative assumption and derive the changes in likelihood given modifications to the forest structure. Previous forests learners have used an additive assumption, e.g. averaging and aggregating, thereby taking advantage of properties of ensembles [12, 13]. However, if we take the sum over the intensity factors from each tree, there are no direct methods for calculating the change in likelihood aside from calculating the likelihood before and after a forest modification, which would require scanning the full data once per modification proposal. Furthermore, summing intensity factors could lead to intensities outside the valid domain $[0, \infty)$.

Instead we use a multiplicative assumption since it gives us the correct range over intensities. As we show below, using the multiplicative assumption also has the advantage that it is easy to compute the change in log likelihood with changes in forest structure. Consider a partition-based CTBN $\mathcal{M} = (\mathcal{B}, \{\mathcal{F}_{x'}\})$ where the partitions $P_{x'}$ and intensities $q_{x'|p}$ are given by the CIFs $\{\mathcal{F}_{x'}\}$. We focus on change in forest structure for one state $x' \in X$ and remove $x'$ from the subscript notation for simplicity. Given a current forest structure $\mathcal{F}$ and its partition $P$, we formulate the change in likelihood by adding a new CIT $f'$ and its partition $P'$. One example of $f'$ is a new a one-split stub. Another example of $f'$ is a tree copied to have the same structure as a CIT $f$ in $\mathcal{F}$ with all intensity factors set to one, except at one leaf node where a split is added. This is equivalent to adding a split to $f$. We denote $\hat{P}$ as the joint partition of $P$ and $P'$ and parts $\hat{p} \in \hat{P}$, $p \in P$, and $p' \in P'$. We consider the change in log likelihood $\Delta LL$ given the new and old models:

$$\Delta LL = (\sum_{\hat{p}} M_{\hat{p}} \log q_{\hat{p}} - q_{\hat{p}} T_{\hat{p}}) - (\sum_{p} M_p \log q_p - q_p T_p)$$

$$= (\sum_{\hat{p}} M_{\hat{p}}(\log q_{p'} + \log q_p) - q_{\hat{p}} T_{\hat{p}}) - (\sum_{p} M_p \log q_p - q_p T_p)$$

$$= (\sum_{\hat{p}} M_{\hat{p}} \log q_{p'} - q_{\hat{p}} T_{\hat{p}}) + \sum_{p} q_p T_p$$

$$= \sum_{p'} M_{p'} \log q_{p'} - \sum_{\hat{p}} q_{\hat{p}} T_{\hat{p}} + \sum_{p} q_p T_p \qquad (4)$$

We make use of the multiplicative assumption that $q_{\hat{p}} = q_{p'} q_p$ and $\sum_p M_p = \sum_{p'} M_{p'} = \sum_{\hat{p}} M_{\hat{p}}$ to arrive at equation 4. The first and third terms are easy to compute given the old intensities and new intensity factors. The second term is slightly more complicated:

$$\sum_{\hat{p}} q_{\hat{p}} T_{\hat{p}} = \sum_{\hat{p}} q_{p'} q_p T_{\hat{p}} = \sum_{p'} q_{p'} \sum_{\hat{p} \sim p'} q_p T_{\hat{p}}$$

We introduce the notation $\hat{p} \sim p'$ to denote the parts $\hat{p}$ that correspond to the part $p'$. The second term is a summation over parts $\hat{p}$; we have simply grouped together terms by membership in $p'$.

The number of parts in the joint partition set $\hat{P}$ can be exponentially large, but the only remaining dependency on the joint partition space in the change in log likelihood is the term $\sum_{\hat{p} \sim p'} q_p T_{\hat{p}}$. We can keep track of this value as we progress through the trajectories, so the actual time cost is linear in the number of trajectory intervals. Thinking of intensities $q$ as rates, and given durations $T$, we observe that the second and third terms in equation 4 are expected numbers of transitions: $E_{\hat{p}} = \sum_{\hat{p}} q_{\hat{p}} T_{\hat{p}}$ and $E_p = \sum_p q_p T_p$. We additionally define $E_{p'} = \sum_{\hat{p} \sim p'} q_p T_{\hat{p}}$. Specifically, the expectations $E_{p'}$ and $E_p$ are the expected number of transitions in part $p'$ and $p$ using the old model intensities, respectively, whereas $E_{\hat{p}}$ is the expected number of transitions using the new intensities.

### 3.4 Maximum-likelihood parameters

The change in log likelihood is dependent on the intensity factor values $\{q_{p'}\}$ we choose for the new partition. We calculate the maximum likelihood parameters by setting the derivative with respect to these factors to zero to get $q_{p'} = \frac{M_{p'}}{\sum_{\hat{p} \sim p'} q_p T_{\hat{p}}} = \frac{M_{p'}}{E_{p'}}$. Following the derivation in [2], we assign priors to the sufficient statistics calculations. Note, however, that the priors affect the multiplicative intensity factors, so a tree may split on the same partition set twice to get a stronger effect on the intensity, with the possible risk of undesirable overfitting.

### 3.5 Forest implementation

We use greedy likelihood maximization steps to learn multiplicative forests (mfCTBNs). Each iteration requires repeating three steps: (re)initialization, sufficient statistics updates, and model updates. Initially we are given a blank forest $\mathcal{F}_{x'}$ per state $x'$ containing a blank tree $f_{x'}$, that is, a single root node acting as a leaf with an intensity factor of one. We also are given sets of possible splits $\{\sigma\}$ and a penalty function $\kappa(|Z|, |\mathcal{M}|)$ to penalize increased model complexity. First, for every leaf $l$ in $\mathcal{M}$, we (re)initialize the sufficient statistics $M_l$ and $E_l$ in $\mathcal{M}$, as well as sufficient statistics for potential forest modifications: $M_{l,\sigma}, E_{l,\sigma}, \forall l, \sigma$. Then, we traverse each of our trajectories $z \in Z$ to update each leaf. For every (state, duration) pair $(s_i, t_i)$, where $t_i$ is the time spent in state $s_{i-1}$ before the transition to $s_i$, we update the sufficient statistics that compose equation 4. Finally, we compute the change in likelihood for possible forest modifications, and choose the modification with the greatest score. If this score is greater than the cost of the additional model complexity, $\kappa$, we accept the modification. We replace the selected leaf with a branch node split upon the selected $\sigma$. The new leaf intensity factors are the product of the old intensity (factor) $q_l$ and the intensity factor $q_{p'}$.

Unlike most forest learning algorithms, mfCTBNs learn trees neither in series nor in parallel. Notably, the best split is determined solely by the change in log likelihood, regardless of the tree to which it belongs. If it belongs to the blank tree at the end of the forest, that tree produces non-trivial factors and a new blank tree is appended to the forest. In this way, as mfCTBN learns, it automatically determines the forest size and tree depth according to the evidence in the data. We provide code and Supplementary Materials at our website.

## 4 Experiments

We evaluate our tree learning and forest learning algorithms on samples from three models. The first model, which we call "Nodelman", is the benchmark model developed in [3, 2]. The second is a simplified cardiovascular health model we call "CV health" shown in Figure 2. The cause of pathologies in this field are known to be multifactorial [14]. For example, it has been well-established that independent positive risk factors for atherosclerosis include being male, a smoker, in old age, having high glucose, high BMI, and high blood pressure. The primary tool for prediction in this field is risk factor analysis, where transformations over the *product* of risk factor values determines overall risk. The third model we call "S100" is a large-scale model with one hundred binary variables. Parents are determined by the binomial distribution $B(0.05, 200)$ over variable states, with

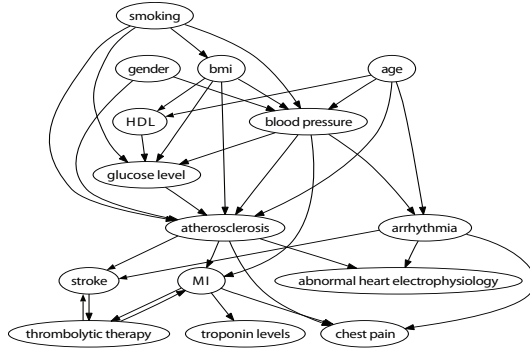

Figure 2: The cardiovascular health (CV health) structure used in experiments.

intensity factor ratios of $1 : 0.5$. Our goal is to show that treeCTBNs and mfCTBNs can scale to much larger model types and still learn effectively. In our experiments we set the potential splits $\{\sigma\}$ to be the set of binary splits determined by indicators for each variable state $x'$. We set $\kappa$ to be zero and terminate model learning when the tune set likelihood begins to decrease.

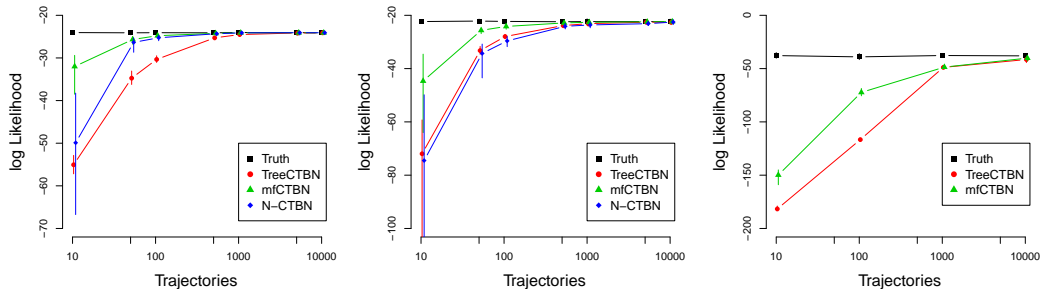

Figure 3: Average testing set log likelihood varying the training set size for each model: Nodelman (left), CV health (center), and S100 (right). N-CTBN averages are omitted on the S100 model as one third of the runs did not terminate.

We compare our algorithms against the learning algorithm presented in [2] using code from [15], which we will call N-CTBN. N-CTBNs perform a greedy Bayesian structure search, adding, removing, or reversing arcs to maximize the Bayesian information criterion score, a tradeoff between the likelihood and a combination of parameter and data size. Our algorithms use a tune set by sieving off one quarter of the original training set trajectories. We use the same Laplace prior as used in [15]. We use the same training and testing set for each algorithm. The trajectories are sampled from the ground truth models for durations $10, 10$ and $2$ units of time, respectively. We evaluate the three models using the testing set average log likelihood. To provide an experimental comparison of model performance, we choose to analyze the p-values for a two-sided paired t-test for the average log likelihoods between mfCTBNs and N-CTBNs for each training set size. The results come from testing sets with one thousand sampled trajectories. Additional evaluation criteria assessing structural recovery were also analyzed and are provided in the Supplementary Materials.

## 4.1 Results

Figure 3 (left) shows that the mfCTBN substantially outperforms both the treeCTBN and the N-CTBN on the Nodelman model in terms of average log likelihood. This effect is most pronounced with relatively few trajectories, suggesting that mfCTBNs are able to learn more quickly than either of the other models.

We observe an even larger difference between the mfCTBN and the other models in the CV health model in Figure 3 (center). With relatively few trajectories, the mfCTBN is able to identify the multifactorial causes as observed in the high log likelihood and structural recall. For runs with fewer than 500 training set trajectories, many N-CTBN models have nodes including every other node as a parent, requiring the estimation of about 300,000 parameters on average, shown in the Supplementary Materials. Figure 3 (right) shows that mfCTBNs can effectively learn dense models an order of magnitude larger than those previously studied. The expected number of parents per node in the S100 model is approximately 20. In order to exactly reconstruct the S100 model, a traditional CTBN would then need to estimate $2^{21}$ intensity values. For many applications, variables need more parents than this. We observe that N-CTBNs have difficulty scaling to models of this size. The N-CTBN learning time on this data set ranges from 4 hours to more than 3 days; runs were stopped if they had not terminated in that time. About one third of the runs failed to complete, and the runs that did complete suggested that N-CTBN performed poorly, similar to the differences observed in the CV health experiment. We suspect the algorithm may be similarly building nodes with many parents; the model might need to estimate $2^{100}$ parameters, a bottleneck at minimum. By comparison, all runs using treeCTBNs and mfCTBNs completed in less than 1 hour. The averaged results of N-CTBNs on the S100 model are omitted accordingly.

We tested for significant differences in the average log likelihoods between the N-CTBN and mfCTBN learning algorithms. In the Nodelman model, the differences were significant at level of $p =$1e-10 for sizes 10 through 500, $p = 0.05$ for sizes 1000 and 5000, and not significant for size 10000. In the CV health model, the differences were significant at $p =$1e-9 for all training set sizes. We were unable to generate a t-test comparison of the S100 model.

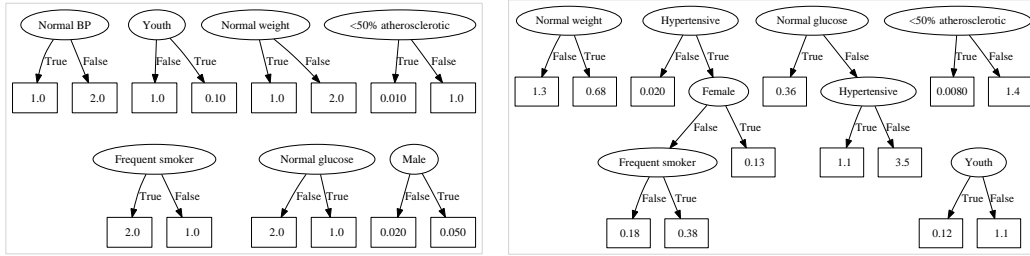

Figure 4: Ground truth (left) and mfCTBN forest learnt from 1000 trajectories (right) for intensity/rate of developing severe atherosclerosis.

Figure 4 shows the ground truth forest and the mfCTBN forest learned for the "severe atherosclerosis" state in the CV health model. To calculate the intensity of transitioning *into* this state, we identify the leaf in each forest that matches the current state and take the product of their intensity factors. Figure 4 (right) shows the recovery of the correct dependencies in approximately the right ratios. Full forest models can be found in the Supplementary Materials.

# 5    Related Work

We discuss the relationships between mfCTBNs and related work in two areas: forest learning and continuous-time processes. Forest learning with a multiplicative assumption is equivalent to forest learning in the log space with an additive assumption and exponentiating the result. This suggests that our method shares similarities with functional gradient boosting (FGB), a leading method for constructing regression forests, run in the log space [16]. However, our method is different in its direct use of a likelihood-based objective function and in its ability to modify any tree in the forest at any iteration. Further discussion comparing the methods is provided in the Supplementary Materials.

Several other works that model variable dependencies over continuous time also exist. Poisson process networks and cascades model variable dependencies and event rates [17, 18]. Perhaps the most closely related work, piecewise-constant conditional intensity models (PCIMs), reframes the concept of a factored CTMP to allow learning over arbitrary basis state functions with trees, possibly piecewise over time [10]. These models focus on the "positive class", i.e. the observation or count of observations of an event. The trouble with this is that the data used to learn the model may be incomplete. Given a timeline, we receive all *observations* of events but not necessarily all *occurrences* of the events, and we would like to include this uncertainty in our model. For Poisson processes in particular, the representation of the "negative" class is missing, when in some cases it is the absent state of a variable that triggers a process, as for example in the case of gene expression networks and negative regulation. Finally other related work includes non-parametric continuous-time processes, which produce exchangeable distributions over transition rate sets in unfactored CTMPs [19].

# 6    Conclusion

We presented an alternative representation of the dynamics of CTBNs using partition-based CTBNs instantiated by trees and forests. Our models grow linearly in the number of forest node splits, while CTBNs grow exponentially in the number of parent nodes per variable. Motivated by the domain over intensities, we introduced multiplicative forests and showed that CTBN likelihood updates can be efficiently computed using changes in log likelihood. Finally, we showed that mfCTBNs outperform both treeCTBNs and N-CTBNs in three experiments and that mfCTBNs are scalable to problems with many variables. With our contributions in developing scalable CTBNs and efficient learning, along with continued improvements in inference, CTBNs can be a powerful statistical tool to model complex processes over continuous time.

# 7    Acknowledgments

We gratefully acknowledge CIBM Training Program grant 5T15LM007359, NIGMS grant R01GM097618-01, NLM grant R01LM011028-01, and ICTR NIH NCATS grant UL1TR000427.

## Footnotes

[1]Note we can generalize this to larger spaces $\mathcal{P} = \mathcal{R} \times \mathcal{X}$, where $\mathcal{R}$ is an external state space as in [10]. but for our analysis we restrict $\mathcal{R}$ to be a single element $r$, i.e. $\mathcal{P} \cong \mathcal{X}$.

[2]Of note, partition-based CTBNs are modeling the intensity of transitioning to the recipient state $x'$, rather than from the donor state $x$ because we are more often interested in the causes of *entering* a state.

# References

[1] T. Dean and K. Kanazawa, "A model for reasoning about persistence and causation," *Computational Intelligence*, vol. 5, no. 2, pp. 142–150, 1989.

[2] U. Nodelman, C. R. Shelton, and D. Koller, "Learning continuous time Bayesian networks," in *UAI*, 2003.

[3] U. Nodelman, *Continuous time Bayesian networks*. PhD thesis, Stanford University, 2007.

[4] U. Nodelman, D. Koller, and C. R. Shelton, "Expectation propagation for continuous time Bayesian networks," in *UAI*, 2005.

[5] S. Saria, U. Nodelman, and D. Koller, "Reasoning at the right time granularity," in *UAI*, 2007.

[6] I. Cohn, T. El-Hay, N. Friedman, and R. Kupferman, "Mean field variational approximation for continuous-time Bayesian networks," in *UAI*, 2009.

[7] Y. Fan and C. R. Shelton, "Sampling for approximate inference in continuous time Bayesian networks," in *AI and Mathematics*, 2008.

[8] V. Rao and Y. Teh, "Fast MCMC sampling for Markov jump processes and continuous time Bayesian networks," in *UAI*, 2011.

[9] D. Heckerman, "Causal independence for knowledge acquisition and inference," in *UAI*, pp. 122–127, 1993.

[10] A. Gunawardana, C. Meek, and P. Xu, "A model for temporal dependencies in event streams," in *NIPS*, 2011.

[11] C. Strobl, J. Malley, and G. Tutz, "An introduction to recursive partitioning: rationale, application, and characteristics of classification and regression trees, bagging, and random forests.," *Psychological methods*, vol. 14, no. 4, p. 323, 2009.

[12] Y. Freund and R. Schapire, "A desicion-theoretic generalization of on-line learning and an application to boosting," in *Computational learning theory*, 1995.

[13] L. Breiman, "Random forests," *Machine learning*, vol. 45, no. 1, pp. 5–32, 2001.

[14] W. Kannel, "Blood pressure as a cardiovascular risk factor," *JAMA*, vol. 275, no. 20, p. 1571, 1996.

[15] C. Shelton, Y. Fan, W. Lam, J. Lee, and J. Xu, "Continuous time Bayesian network reasoning and learning engine," *JMLR*, vol. 11, pp. 1137–1140, 2010.

[16] J. Friedman, "Greedy function approximation: a gradient boosting machine," *Annals of Statistics*, 2001.

[17] S. Rajaram, T. Graepel, and R. Herbrich, "Poisson-networks: A model for structured point processes," in *AI and Statistics*, 2005.

[18] A. Simma, *Modeling Events in Time Using Cascades Of Poisson Processes*. PhD thesis, EECS Department, University of California, Berkeley, Jul 2010.

[19] A. Saeedi and A. Bouchard-Ct, "Priors over recurrent continuous time processes," in *NIPS*, 2011.

